# Convolution Kernels for Natural Language

**Michael Collins**
AT&T Labs–Research
180 Park Avenue, New Jersey, NJ 07932
mcollins@research.att.com

**Nigel Duffy**
Department of Computer Science
University of California at Santa Cruz
nigeduff@cse.ucsc.edu

## Abstract

We describe the application of kernel methods to Natural Language Processing (NLP) problems. In many NLP tasks the objects being modeled are strings, trees, graphs or other discrete structures which require some mechanism to convert them into feature vectors. We describe kernels for various natural language structures, allowing rich, high dimensional representations of these structures. We show how a kernel over trees can be applied to parsing using the voted perceptron algorithm, and we give experimental results on the ATIS corpus of parse trees.

## 1  Introduction

Kernel methods have been widely used to extend the applicability of many well-known algorithms, such as the Perceptron [1], Support Vector Machines [6], or Principal Component Analysis [15]. A key property of these algorithms is that the only operation they require is the evaluation of dot products between pairs of examples. One may therefore replace the dot product with a Mercer kernel, implicitly mapping feature vectors in $\mathbb{R}^d$ into a new space $\mathbb{R}^n$, and applying the original algorithm in this new feature space. Kernels provide an efficient way to carry out these calculations when $n$ is large or even infinite.

This paper describes the application of kernel methods to Natural Language Processing (NLP) problems. In many NLP tasks the input domain cannot be neatly formulated as a subset of $\mathbb{R}^d$. Instead, the objects being modeled are strings, trees or other discrete structures which require some mechanism to convert them into feature vectors. We describe kernels for various NLP structures, and show that they allow computationally feasible representations in very high dimensional feature spaces, for example a parse tree representation that tracks all subtrees. We show how a tree kernel can be applied to parsing using the perceptron algorithm, giving experimental results on the ATIS corpus of parses. The kernels we describe are instances of "Convolution Kernels", which were introduced by Haussler [10] and Watkins [16], and which involve a recursive calculation over the "parts" of a discrete structure. Although we concentrate on NLP tasks in this paper, the kernels should also be useful in computational biology, which shares similar problems and structures.

### 1.1  Natural Language Tasks

Figure 1 shows some typical structures from NLP tasks. Each structure involves an "observed" string (a sentence), and some hidden structure (an underlying state sequence or tree). We assume that there is some training set of structures, and that the task is to learn

a) Lou Gerstner is chairman of IBM $\rightarrow$
     [S [NP Lou Gerstner ] [VP is [NP chairman [PP of [NP IBM ] ] ] ] ]

b) Lou Gerstner is chairman of IBM $\rightarrow$ Lou/SP Gerstner/CP is/N chairman/N of/N IBM/SC
c) Lou/N Gerstner/N is/V chairman/N of/P IBM/N

Figure 1: Three NLP tasks where a function is learned from a string to some hidden structure. In (a), the hidden structure is a parse tree. In (b), the hidden structure is an underlying sequence of states representing named entity boundaries (SP = Start person, CP = Continue person, SC = Start company, N= No entity). In (c), the hidden states represent part-of-speech tags (N = noun, V = verb, P = preposition,).

the mapping from an input string to its hidden structure. We refer to tasks that involve trees as *parsing* problems, and tasks that involve hidden state sequences as *tagging* problems.

In many of these problems ambiguity is the key issue: although only one analysis is plausible, there may be very many *possible* analyses. A common way to deal with ambiguity is to use a stochastic grammar, for example a Probabilistic Context Free Grammar (PCFG) for parsing, or a Hidden Markov Model (HMM) for tagging. Probabilities are attached to rules in the grammar – context-free rules in the case of PCFGs, state transition probabilities and state emission probabilities for HMMs. Rule probabilities are typically estimated using maximum likelihood estimation, which gives simple relative frequency estimates. Competing analyses for the same sentence are ranked using these probabilities. See [3] for an introduction to these methods.

This paper proposes an alternative to generative models such as PCFGs and HMMs. Instead of identifying parameters with rules of the grammar, we show how kernels can be used to form representations that are sensitive to larger sub-structures of trees or state sequences. The parameter estimation methods we describe are discriminative, optimizing a criterion that is directly related to error rate.

While we use the parsing problem as a running example in this paper, kernels over NLP structures could be used in many ways: for example, in PCA over discrete structures, or in classification and regression problems. Structured objects such as parse trees are so prevalent in NLP that convolution kernels should have many applications.

## 2   A Tree Kernel

The previous section introduced PCFGs as a parsing method. This approach essentially counts the relative number of occurences of a given rule in the training data and uses these counts to represent its learned knowledge. PCFGs make some fairly strong independence assumptions, disregarding substantial amounts of structural information. In particular, it does not appear reasonable to assume that the rules applied at level $i$ in the parse tree are unrelated to those applied at level $i + 1$.

As an alternative we attempt to capture considerably more structural information by considering all tree fragments that occur in a parse tree. This allows us to capture higher order dependencies between grammar rules. See figure 2 for an example. As in a PCFG the new representation tracks the counts of single rules, but it is also sensitive to larger sub-trees.

Conceptually we begin by enumerating all tree fragments that occur in the training data $1, \ldots, n$. Note that this is done only implicitly. Each tree is represented by an $n$ dimensional vector where the $i$'th component counts the number of occurences of the $i$'th tree fragment. Let us define the function $h_i(T)$ to be the number of occurences of the $i$'th tree fragment in tree $T$, so that $T$ is now represented as $\mathbf{h}(T) = (h_1(T), h_2(T), \ldots, h_n(T))$.

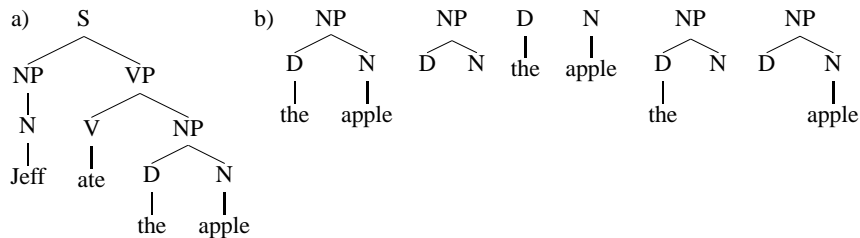

Figure 2: a) An example tree. b) The sub-trees of the NP covering *the apple*. The tree in (a) contains all of these sub-trees, and many others. We define a sub-tree to be any subgraph which includes more than one node, with the restriction that entire (not partial) rule productions must be included. For example, the fragment [NP [D the ]] is excluded because it contains only part of the production NP → D N.

Note that $n$ will be huge (a given tree will have a number of subtrees that is exponential in its size). Because of this we would like design algorithms whose computational complexity does not depend on $n$.

Representations of this kind have been studied extensively by Bod [2]. However, the work in [2] involves training and decoding algorithms that depend computationally on the number of subtrees involved.[1] The parameter estimation techniques described in [2] do not correspond to maximum-likelihood estimation or a discriminative criterion: see [11] for discussion. The methods we propose show that the score for a parse can be calculated in polynomial time in spite of an exponentially large number of subtrees, and that efficient parameter estimation techniques exist which optimize discriminative criteria that have been well-studied theoretically.

Goodman [9] gives an ingenious conversion of the model in [2] to an equivalent PCFG whose number of rules is linear in the size of the training data, thus solving many of the computational issues. An exact implementation of Bod's parsing method is still infeasible, but Goodman gives an approximation that can be implemented efficiently. However, the method still suffers from the lack of justification of the parameter estimation techniques.

The key to our efficient use of this high dimensional representation is the definition of an appropriate kernel. We begin by examining the inner product between two trees $T_1$ and $T_2$ under this representation, $K(T_1, T_2) = \mathbf{h}(T_1) \cdot \mathbf{h}(T_2)$. To compute $K$ we first define the set of nodes in trees $T_1$ and $T_2$ as $N_1$ and $N_2$ respectively. We define the indicator function $I_i(n)$ to be 1 if sub-tree $i$ is seen rooted at node $n$ and 0 otherwise. It follows that $h_i(T_1) = \sum_{n_1 \in N_1} I_i(n_1)$ and $h_i(T_2) = \sum_{n_2 \in N_2} I_i(n_2)$. The first step to efficient computation of the inner product is the following property (which can be proved with some simple algebra):

$$\mathbf{h}(T_1) \cdot \mathbf{h}(T_2) = \sum_i h_i(T_1) h_i(T_2) = \sum_{n_1 \in N_1} \sum_{n_2 \in N_2} \sum_i I_i(n_1) I_i(n_2) = \sum_{n_1 \in N_1} \sum_{n_2 \in N_2} C(n_1, n_2)$$

where we define $C(n_1, n_2) = \sum_i I_i(n_1) I_i(n_2)$. Next, we note that $C(n_1, n_2)$ can be computed in polynomial time, due to the following recursive definition:

• If the productions at $n_1$ and $n_2$ are different $C(n_1, n_2) = 0$.

• If the productions at $n_1$ and $n_2$ are the same, and $n_1$ and $n_2$ are pre-terminals, then $C(n_1, n_2) = 1$.[2]

- Else if the productions at $n_1$ and $n_2$ are the same and $n_1$ and $n_2$ are not pre-terminals,

$$C(n_1, n_2) = \prod_{j=1}^{nc(n_1)} (1 + C(ch(n_1, j), ch(n_2, j))) \,,$$

where $nc(n_1)$ is the number of children of $n_1$ in the tree; because the productions at $n_1/n_2$ are the same, we have $nc(n_1) = nc(n_2)$. The $i$'th child-node of $n_1$ is $ch(n_1, i)$.

To see that this recursive definition is correct, note that $C(n_1, n_2)$ simply counts the number of *common subtrees* that are found rooted at both $n_1$ and $n_2$. The first two cases are trivially correct. The last, recursive, definition follows because a common subtree for $n_1$ and $n_2$ can be formed by taking the production at $n_1/n_2$, together with a choice at each child of simply taking the non-terminal at that child, or any one of the common sub-trees at that child. Thus there are $(1 + C(child(n_1, i), child(n_2, i)))$ possible choices at the $i$'th child. (Note that a similar recursion is described by Goodman [9], Goodman's application being the conversion of Bod's model [2] to an equivalent PCFG.)

It is clear from the identity $\mathbf{h}(T_1) \cdot \mathbf{h}(T_2) = \sum_{n_1, n_2} C(n_1, n_2)$, and the recursive definition of $C(n_1, n_2)$, that $\mathbf{h}(T_1) \cdot \mathbf{h}(T_2)$ can be calculated in $O(|N_1||N_2|)$ time: the matrix of $C(n_1, n_2)$ values can be filled in, then summed. This can be a pessimistic estimate of the runtime. A more useful characterization is that it runs in time linear in the number of members $(n_1, n_2) \in N_1 \times N_2$ such that the productions at $n_1$ and $n_2$ are the same. In our data we have found a typically linear number of nodes with identical productions, so that most values of $C$ are 0, and the running time is close to linear in the size of the trees.

This recursive kernel structure, where a kernel between two objects is defined in terms of kernels between its parts is quite a general idea. Haussler [10] goes into some detail describing which construction operations are valid in this context, i.e. which operations maintain the essential Mercer conditions. This paper and previous work by Lodhi et al. [12] examining the application of convolution kernels to strings provide some evidence that convolution kernels may provide an extremely useful tool for applying modern machine learning techniques to highly structured objects. The key idea here is that one may take a structured object and split it up into parts. If one can construct kernels over the parts then one can combine these into a kernel over the whole object. Clearly, this idea can be extended recursively so that one only needs to construct kernels over the "atomic" parts of a structured object. The recursive combination of the kernels over parts of an object retains information regarding the structure of that object.

Several issues remain with the kernel we describe over trees and convolution kernels in general. First, the value of $K(T_1, T_2)$ will depend greatly on the size of the trees $T_1, T_2$. One may normalize the kernel by using $K'(T_1, T_2) = K(T_1, T_2)/\sqrt{K(T_1, T_1)K(T_2, T_2)}$ which also satisfies the essential Mercer conditions. Second, the value of the kernel when applied to two copies of the same tree can be extremely large (in our experiments on the order of $10^6$) while the value of the kernel between two different trees is typically much smaller (in our experiments the typical pairwise comparison is of order 100). By analogy with a Gaussian kernel we say that the kernel is very peaked. If one constructs a model which is a linear combination of trees, as one would with an SVM [6] or the perceptron, the output will be dominated by the most similar tree and so the model will behave like a nearest neighbor rule. There are several possible solutions to this problem. Following Haussler [10] we may radialize the kernel, however, it is not always clear that the result is still a valid kernel. Radializing did not appear to help in our experiments.

These problems motivate two simple modifications to the tree kernel. Since there will be many more tree fragments of larger size – say depth four versus depth three – and

---

symbols in Figure 2.

consequently less training data, it makes sense to downweight the contribution of larger tree fragments to the kernel. The first method for doing this is to simply restrict the depth of the tree fragments we consider.[3] The second method is to scale the relative importance of tree fragments with their size. This can be achieved by introducing a parameter $0 < \lambda \leq 1$, and modifying the base case and recursive case of the definitions of $C$ to be respectively

$$C(n_1, n_2) = \lambda \quad \text{and} \quad C(n_1, n_2) = \lambda \prod_{j=1}^{nc(n_1)} (1 + C(ch(n_1, j), ch(n_2, j))) \; .$$

This corresponds to a modified kernel $\mathbf{h}(T_1) \cdot \mathbf{h}(T_2) = \sum_i \lambda^{size_i} h_i(T_1) h_i(T_2)$, where $size_i$ is the number of rules in the $i$'th fragment. This kernel downweights the contribution of tree fragments exponentially with their size.

It is straightforward to design similar kernels for tagging problems (see figure 1) and for another common structure found in NLP, dependency structures. See [5] for details. In the tagging kernel, the implicit feature representation tracks all features consisting of a subsequence of state labels, each with or without an underlying word. For example, the paired sequence {Lou/SP Gerstner/CP is/N chairman/N of/N IBM/SC} would include features such as {SP CP}, {SP Gerstner/CP N}, {SP CP is/N N of/N} and so on.

## 3   Linear Models for Parsing and Tagging

This section formalizes the use of kernels for parsing and tagging problems. The method is derived by the transformation from ranking problems to a margin-based classification problem in [8]. It is also related to the Markov Random Field methods for parsing suggested in [13], and the boosting methods for parsing in [4]. We consider the following set-up:

• Training data is a set of example input/output pairs. In parsing we would have training examples $\{s_i, t_i\}$ where each $s_i$ is a sentence and each $t_i$ is the correct tree for that sentence.

• We assume some way of enumerating a set of candidates for a particular sentence. We use $\mathbf{x}_{ij}$ to denote the $j$'th candidate for the $i$'th sentence in training data, and $\mathcal{C}(s_i) = \{\mathbf{x}_{i1}, \mathbf{x}_{i2} \ldots\}$ to denote the set of candidates for $s_i$. [4]

• Without loss of generality we take $\mathbf{x}_{i1}$ to be the correct parse for $s_i$ (i.e., $\mathbf{x}_{i1} = t_i$).

• Each candidate $\mathbf{x}_{ij}$ is represented by a feature vector $\mathbf{h}(\mathbf{x}_{ij})$ in the space $\mathbb{R}^n$. The parameters of the model are also a vector $\bar{w} \in \mathbb{R}^n$. We then define the "ranking score" of each example as $\bar{w} \cdot \mathbf{h}(\mathbf{x}_{ij})$. This score is interpreted as an indication of the plausibility of the candidate. The output of the model on a training or test example $s$ is $\operatorname{argmax}_{\mathbf{x} \in \mathcal{C}(s)} \bar{w} \cdot \mathbf{h}(\mathbf{x})$.

When considering approaches to training the parameter vector $\bar{w}$, note that a ranking function that correctly ranked the correct parse above all competing candidates would satisfy the conditions $\bar{w} \cdot (\mathbf{h}(\mathbf{x}_{i1}) - \mathbf{h}(\mathbf{x}_{ij})) > 0 \; \forall i, \; \forall j \geq 2$. It is simple to modify the Perceptron and Support Vector Machine algorithms to treat this problem. For example, the SVM optimization problem (hard margin version) is to find the $\bar{w}^*$ which minimizes $\|\bar{w}\|^2$ subject to the constraints $\bar{w} \cdot (\mathbf{h}(\mathbf{x}_{i1}) - \mathbf{h}(\mathbf{x}_{ij})) \geq 1 \; \forall i, \; \forall j \geq 2$. Rather than explicitly calculating $\bar{w}$, the perceptron algorithm and Support Vector Machines can be formulated as a search

**Define:** $F(\mathbf{x}) = \sum_{(i,j)} \alpha_{i,j} \left( \mathbf{h}(\mathbf{x}_{i1}) \cdot \mathbf{h}(\mathbf{x}) - \mathbf{h}(\mathbf{x}_{ij}) \cdot \mathbf{h}(\mathbf{x}) \right)$
**Initialization:** Set dual parameters $\alpha_{i,j} = 0$
**For** $i = 1 \ldots n, \;\; j = 2 \ldots n_i$
   **If** $F(\mathbf{x}_{i1}) > F(\mathbf{x}_{ij})$ do nothing, **Else** $\alpha_{ij} = \alpha_{ij} + 1$

Figure 3: The perceptron algorithm for ranking problems.

| Depth | 1 | 2 | 3 | 4 | 5 | 6 |
|---|---|---|---|---|---|---|
| Score | $73 \pm 1$ | $79 \pm 1$ | $80 \pm 1$ | $79 \pm 1$ | $79 \pm 1$ | $78 \pm 0.01$ |
| Improvement | $-1 \pm 4$ | $20 \pm 6$ | $23 \pm 3$ | $21 \pm 4$ | $19 \pm 4$ | $18 \pm 3$ |

Table 1: *Score* shows how the parse score varies with the maximum depth of sub-tree considered by the perceptron. *Improvement* is the relative reduction in error in comparison to the PCFG, which scored 74%. The numbers reported are the mean and standard deviation over the 10 development sets.

for "dual parameters" $\alpha_{ij}$ which determine the optimal weights $\bar{w}^*$

$$\bar{w}^* = \sum_{(i,j)} \alpha_{i,j} \left( \mathbf{h}(\mathbf{x}_{i1}) - \mathbf{h}(\mathbf{x}_{ij}) \right) \tag{1}$$

(we use $\sum_{(i,j)}$ as shorthand for $\sum_i \sum_{j=2}^{n_i}$). It follows that the score of a parse can be calculated using the dual parameters, and inner products between feature vectors, without having to explicitly deal with feature or parameter vectors in the space $\mathbb{R}^n$:

$$\bar{w}^* \cdot \mathbf{x} = \sum_{(i,j)} \alpha_{i,j} \left( \mathbf{h}(\mathbf{x}_{i1}) \cdot \mathbf{h}(\mathbf{x}) - \mathbf{h}(\mathbf{x}_{ij}) \cdot \mathbf{h}(\mathbf{x}) \right)$$

For example, see figure 3 for the perceptron algorithm applied to this problem.

## 4  Experimental Results

To demonstrate the utility of convolution kernels for natural language we applied our tree kernel to the problem of parsing the Penn treebank ATIS corpus [14]. We split the treebank randomly into a training set of size 800, a development set of size 200 and a test set of size 336. This was done 10 different ways to obtain statistically significant results. A PCFG was trained on the training set, and a beam search was used to give a set of parses, with PCFG probabilities, for each of the sentences. We applied a variant of the voted perceptron algorithm [7], which is a more robust version of the original perceptron algorithm with performance similar to that of SVMs. The voted perceptron can be kernelized in the same way that SVMs can but it can be considerably more computationally efficient.

We generated a ranking problem by having the PCFG generate its top 100 candidate parse trees for each sentence. The voted perceptron was applied, using the tree kernel described previously, to this re-ranking problem. It was trained on 20 trees selected randomly from the top 100 for each sentence and had to choose the best candidate from the top 100 on the test set. We tested the sensitivity to two parameter settings: first, the maximum depth of sub-tree examined, and second, the scaling factor used to down-weight deeper trees. For each value of the parameters we trained on the training set and tested on the development set. We report the results averaged over the development sets in Tables 1 and 2.

We report a parse score which combines precision and recall. Define $c_i$ to be the number of correctly placed constituents in the $i$'th test tree, $p_i$ to be the number of constituents

| Scale | 0.1 | 0.2 | 0.3 | 0.4 | 0.5 | 0.6 | 0.7 | 0.8 | 0.9 |
|---|---|---|---|---|---|---|---|---|---|
| Score | $77 \pm 1$ | $78 \pm 1$ | $79 \pm 1$ | $79 \pm 1$ | $79 \pm 1$ | $79 \pm 1$ | $79 \pm 1$ | $79 \pm 1$ | $78 \pm 1$ |
| Imp. | $11 \pm 6$ | $17 \pm 5$ | $20 \pm 4$ | $21 \pm 3$ | $21 \pm 4$ | $22 \pm 4$ | $21 \pm 4$ | $19 \pm 4$ | $17 \pm 5$ |

Table 2: *Score* shows how the parse score varies with the scaling factor for deeper sub-trees is varied. *Imp.* is the relative reduction in error in comparison to the PCFG, which scored 74%. The numbers reported are the mean and standard deviation over the 10 development sets.

proposed, and $g_i$ to be the number of constituents in the true parse tree. A constituent is defined by a non-terminal label and its span. The score is then

$$100\% * \frac{1}{\sum_i g_i} \sum_i g_i \times \frac{1}{2} \left( \frac{c_i}{p_i} + \frac{c_i}{g_i} \right)$$

The precision and recall on the $i$'th parse are $c_i/p_i$ and $c_i/g_i$ respectively. The score is then the average precision recall, weighted by the size of the trees $g_i$. We also give relative improvements over the PCFG scores. If the PCFG score is $x$ and the perceptron score is $y$, the relative improvement is $100\% * (y - x)/(100 - x)$, i.e., the relative reduction in error.

We finally used the development set for cross-validation to choose the best parameter settings for each split. We used the best parameter settings (on the development sets) for each split to train on both the training and development sets, then tested on the test set. This gave a relative goodness score of $80\% \pm 1$ with the best choice of maximum depth and a score of $80\% \pm 1$ with the best choice of scaling factor. The PCFG scored 74% on the test data. All of these results were obtained by running the perceptron through the training data only once. As has been noted previously by Freund and Schapire [7], the voted perceptron often obtains better results when run multiple times through the training data. Running through the data twice with a maximum depth of 3 yielded a relative goodness score of $81\% \pm 1$, while using a larger number of iterations did not improve the results significantly.

In summary we observe that in these simple experiments the voted perceptron and an appropriate convolution kernel can obtain promising results. However there are other methods which perform considerably better than a PCFG for NLP parsing – see [3] for an overview – future work will investigate whether the kernels in this paper give performance gains over these methods.

## 5   A Compressed Representation

When used with algorithms such as the perceptron, convolution kernels may be even more computationally attractive than the traditional radial basis or polynomial kernels. The linear combination of parse trees constructed by the perceptron algorithm can be viewed as a weighted forest. One may then search for subtrees in this weighted forest that occur more than once. Given a linear combination of two trees $aT_1 + bT_2$ which contain a common subtree, we may construct a smaller weighted acyclic graph, in which the common subtree occurs only once and has weight $a+b$. This process may be repeated until an arbitrary linear combination of trees is collapsed into a weighted acyclic graph in which no subtree occurs more than once. The perceptron may now be evaluated on a new tree by a straightforward generalization of the tree kernel to weighted acyclic graphs of the form produced by this procedure.

Given the nature of our data – the parse trees have a high branching factor, the words are chosen from a dictionary that is relatively small in comparison to the size of the training data, and are drawn from a very skewed distribution, and the ancestors of leaves are part

of speech tags – there are a relatively small number of subtrees in the lower levels of the parse trees that occur frequently and make up the majority of the data. It appears that the approach we have described above should save a considerable amount of computation. This is something we intend to explore further in future work.

## 6  Conclusions

In this paper we described how convolution kernels can be used to apply standard kernel based algorithms to problems in natural language. Tree structures are ubiquitous in natural language problems and we illustrated the approach by constructing a convolution kernel over tree structures. The problem of parsing English sentences provides an appealing example domain and our experiments demonstrate the effectiveness of kernel-based approaches to these problems. Convolution kernels combined with such techniques as kernel PCA and spectral clustering may provide a computationally attractive approach to many other problems in natural language processing. Unfortunately, we are unable to expand on the many potential applications in this short note, however, many of these issues are spelled out in a longer Technical Report [5].

## Footnotes

[1] In training, a parameter is explicitly estimated for each sub-tree. In searching for the best parse, calculating the score for a parse in principle requires summing over an exponential number of derivations underlying a tree, and in practice is approximated using Monte-Carlo techniques.

[2] Pre-terminals are nodes directly above words in the surface string, for example the N, V, and D

[3]This can be achieved using a modified dynamic programming table where $C(n_1, n_2, d)$ stores the number of common subtrees at nodes $n_1, n_2$ of depth $d$ or less. The recursive definition of $C$ can be modified appropriately.

[4]A context-free grammar – perhaps taken straight from the training examples – is one way of enumerating candidates. Another choice is to use a hand-crafted grammar (such as the LFG grammar in [13]) or to take the $n$ most probable parses from an existing probabilistic parser (as in [4]).

## References

[1]  Aizerman, M., Braverman, E., and Rozonoer, L. (1964). Theoretical Foundations of the Potential Function Method in Pattern Recognition Learning. *Automation and Remote Control*, 25:821–837.

[2]  Bod, R. (1998). *Beyond Grammar: An Experience-Based Theory of Language*. CSLI Publications/Cambridge University Press.

[3]  Charniak, E. (1997). Statistical techniques for natural language parsing. In *AI Magazine*, Vol. 18, No. 4.

[4]  Collins, M. (2000). Discriminative Reranking for Natural Language Parsing. *Proceedings of the Seventeenth International Conference on Machine Learning*. San Francisco: Morgan Kaufmann.

[5]  Collins, M. and Duffy, N. (2001). Parsing with a Single Neuron: Convolution Kernels for Natural Language Problems. Technical report UCSC-CRL-01-01, University of California at Santa Cruz.

[6]  Cortes, C. and Vapnik, V. (1995). Support–Vector Networks. *Machine Learning*, 20(3):273–297.

[7]  Freund, Y. and Schapire, R. (1999). Large Margin Classification using the Perceptron Algorithm. In *Machine Learning*, 37(3):277–296.

[8]  Freund, Y., Iyer, R.,Schapire, R.E., & Singer, Y. (1998). An efficient boosting algorithm for combining preferences. In *Machine Learning: Proceedings of the Fifteenth International Conference*. San Francisco: Morgan Kaufmann.

[9]  Goodman, J. (1996). Efficient algorithms for parsing the DOP model. In *Proceedings of the Conference on Empirical Methods in Natural Language Processing (EMNLP 96)*, pages 143-152.

[10]  Haussler, D. (1999). *Convolution Kernels on Discrete Structures.* Technical report, University of Santa Cruz.

[11]  Johnson, M. The DOP estimation method is biased and inconsistent. To appear in *Computational Linguistics*.

[12]  Lodhi, H., Christianini, N., Shawe-Taylor, J., and Watkins, C. (2001). Text Classification using String Kernels. To appear in *Advances in Neural Information Processing Systems 13*, MIT Press.

[13]  Johnson, M., Geman, S., Canon, S., Chi, S., & Riezler, S. (1999). Estimators for stochastic 'unification-based" grammars. In *Proceedings of the 37th Annual Meeting of the Association for Computational Linguistics*. San Francisco: Morgan Kaufmann.

[14]  Marcus, M., Santorini, B., & Marcinkiewicz, M. (1993). Building a large annotated corpus of english: The Penn treebank. *Computational Linguistics, 19*, 313-330.

[15]  Scholkopf, B., Smola, A.,and Muller, K.-R. (1999). Kernel principal component analysis. In B. Scholkopf, C. J. C. Burges, and A. J. Smola, editors, Advances in Kernel Methods – SV Learning, pages 327-352. MIT Press, Cambridge, MA.

[16]  Watkins, C. (2000). Dynamic alignment kernels. In A.J. Smola, P.L. Bartlett, B. Schlkopf, and D. Schuurmans, editors, Advances in Large Margin Classifiers, pages 39-50, MIT Press.
